# Bounded Finite State Controllers

**Pascal Poupart**
Department of Computer Science
University of Toronto
Toronto, ON M5S 3H5
ppoupart@cs.toronto.edu

**Craig Boutilier**
Department of Computer Science
University of Toronto
Toronto, ON M5S 3H5
cebly@cs.toronto.edu

## Abstract

We describe a new approximation algorithm for solving partially observable MDPs. Our *bounded policy iteration* approach searches through the space of bounded-size, stochastic finite state controllers, combining several advantages of gradient ascent (efficiency, search through restricted controller space) and policy iteration (less vulnerability to local optima).

## 1   Introduction

Finite state controllers (FSCs) provide a simple, convenient way of representing policies for partially observable Markov decision processes (POMDPs). Two general approaches are often used to construct good controllers: policy iteration (PI) [7] and gradient ascent (GA) [10, 11, 1]. The former is guaranteed to converge to an optimal policy, however, the size of the controller often grows intractably. In contrast, the latter restricts its search to controllers of a bounded size, but may get trapped in a local optimum.

While locally optimal solutions are often acceptable, for many planning problems with a combinatorial flavor, GA can easily get trapped by simple policies that are far from optimal. Consider a system engaged in preference elicitation, charged with discovering optimal query policy to determine relevant aspects of a user's utility function. Often no single question yields information of much value, while a sequence of queries does. If each question has a cost, a system that locally optimizes the policy by GA may determine that the best course of action is to ask no questions (i.e., minimize cost given no information gain). When an optimal policy consists of a sequence of actions any small perturbation to which results in a bad policy, there is little hope of finding this sequence using methods that greedily perform local perturbations such as those employed by GA.

In general, we would like the best of both worlds: bounded controller size and convergence to a global optimum. While achieving both is NP-hard for the class of deterministic controllers [10], one can hope for a tractable algorithm that at least avoids obvious local optima. We propose a new anytime algorithm, *bounded policy iteration (BPI)* that improves a policy much like Hansen's PI [7] while keeping the size of the controller fixed. Whenever the algorithm gets stuck in a local optimum, the controller is allowed to slightly grow by introducing one (or a few) node(s) to escape the local optimum.

Following a brief review of FSCs (Sec. 2), we extend PI to stochastic controllers (Sec. 3), thus admitting smaller, high quality controllers. We then derive the BPI algorithm by ensuring that the number of nodes remains unchanged (Sec. 4). We analyze the structure of

local optima for BPI (Sec. 5), relate this analysis to GA, and use it to justify a new method to escape local optima. Finally, we report some preliminary experiments (Sec. 6).

## 2  Finite State Controllers for POMDPs

A POMDP is defined by a set of states $\mathcal{S}$; a set of actions $\mathcal{A}$; a set of observations $\mathcal{Z}$; a transition function $T$, where $T(s, a, s')$ denotes the transition probabilities $Pr(s'|s, a)$; an observation function $Z$, where $Z(s, z)$ denotes the probability $Pr(z|s, a)$ of making observation $z$ in state $s$ after taking action $a$; and a reward function $R$, where $R(s, a)$ denotes the immediate reward associated with state $s$ when executing ation $a$. We assume discrete state, action and observation sets and we focus on discounted, infinite horizon POMDPs with discount factor $0 \leq \gamma < 1$. Since states are not directly observable in POMDPs, we define a belief state $b(s) = Pr(s)$ to be a distribution over states. Belief state $b$ can be updated in response to a action-observation pair $\langle a, z \rangle$ using Bayes rule.

Policies represented by FSCs are defined by a (possibly cyclic) directed graph $\pi = \langle \mathcal{N}, \mathcal{E} \rangle$, where each node $n \in \mathcal{N}$ is labeled by an action $a$ and each edge $e \in \mathcal{E}$ by an observation $z$. Each node has one outward edge per observation. The FSC can be viewed as a policy $\pi = \langle \alpha, \beta \rangle$, where *action strategy* $\alpha$ associates each node $n$ with an action $\alpha(n) \in \mathcal{A}$, and *observation strategy* $\beta$ associates each node $n$ and observation $z$ with a successor node $\beta(n, z) \in \mathcal{N}$ (corresponding to the edge from $n$ labeled with $z$). A policy is executed by taking the action associated with the "current node," and updating the current node by following the edge labeled by the observation made.

The value function $V^\pi$ of an FSC $\pi$ is the expected discounted sum of rewards for executing its policy $\pi$, and can be computed by solving a set of linear equations:

$$V^\pi(n, s) = R(s, \alpha(n)) + \gamma \sum_z Pr(s'|s, \alpha(n)) Pr(z|s', \alpha(n)) V^\pi(\beta(n, z), s') \quad (1)$$

Given an initial belief state $b$, an FSC's value at node $n$ is simply the expectation $V(n, b) = \sum_s b(s) V(n, s)$; the best starting node for a given $b$ is determined by $V(b) = \max_n V(n, b)$. As a result, the value $V(n, b)$ of each node $n$ is linear with respect to the belief state; hence the value function of the controller is piecewise-linear and convex. In Fig. 1(a), each linear segment corresponds to the value function of a node and the upper surface of these segments forms the controller value function. The optimal value function $V^*$ satisfies Bellman's equation:

$$V^*(b) = \max_a R(b, a) + \gamma \sum_z Pr(z|b, a) V(b_z^a) \quad (2)$$

Policy iteration (PI) [7] incrementally improves a controller by alternating between two steps, policy improvement and policy evaluation, until convergence to an optimal policy. Policy evaluation solves Eq. 1 for a given policy. Policy improvement adds nodes to the controller by dynamic programming (DP) and removes other nodes. A DP backup applies the r.h.s. of Eq. 2 to the value function ($V$ in Fig. 2(a)) of the current controller to obtain a new, improved value function ($V'$ in Fig. 2(a)). Each linear segment of $V'$ corresponds to a new node added to the controller. Several algorithms can be used to perform DP backups, with incremental pruning [4] perhaps being the fastest. After the new nodes created by DP have been added, old nodes that are now *pointwise dominated* are removed. A node is pointwise dominated when its value is less than that of some other node at all belief states (e.g., $n_1$ is pointwise dominated by $n_4$ in Fig. 2(a)). The inward edges of a pointwise dominated node are re-directed to the dominating node since it offers better value (e.g., inward arcs of $n_1$ are redirected to $n_4$ in Fig. 2(c)). The controller resulting from this policy improvement step is guaranteed to offer higher value at all belief states. On the other hand, up to $|\mathcal{A}||\mathcal{N}|^{|\mathcal{Z}|}$ new nodes may be added with each DP backup, so the size of the controller quickly becomes intractable in many POMDPs.

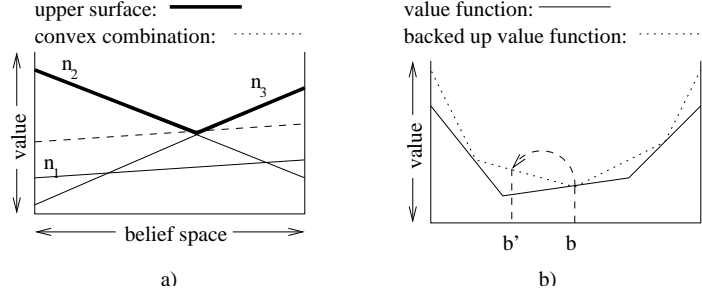

Figure 1: a) Value function example; b) BPI local optimum: each linear segment of the value function is tangent to the backed up value function

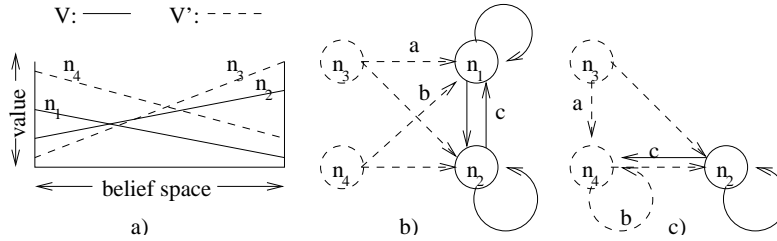

Figure 2: a) Value function $V$ and the backed-up $V'$ obtained by DP; b) original controller ($n_1$ and $n_2$) with nodes added ($n_3$ and $n_4$) by DP; c) new controller once pointwise dominated node $n_1$ is removed and its inward arcs a, b, c are redirected to $n_4$

## 3   Policy Iteration for Stochastic Controllers

Policy iteration only prunes nodes that are pointwise dominated, rather than all dominated nodes. This is because the algorithm is designed to produce controllers with deterministic observation strategies. A pointwise-dominated node can safely be pruned since its inward arcs are redirected to the dominating node (which has value at least as high as the dominated node at each state). In contrast, a node jointly dominated by several nodes (e.g., $n_2$ in Fig. 2(b) is jointly dominated by $n_3$ and $n_4$) cannot be pruned without its inward arcs being redirected to different nodes depending on the current belief state.

This problem can be circumvented by allowing stochastic observation strategies. We revise the notion of observation strategy $\beta(n, z, n') = Pr(n'|n, z)$, defining a distribution over successor nodes $n'$ for each $n, z$-pair. If the stochastic strategy is chosen carefully, the corresponding convex combination of dominating nodes may pointwise dominate the node we would like to prune. In Fig. 1(a), $n_1$ is dominated by $n_2$ and $n_3$ together (but neither of them alone). Convex combinations of $n_2$ and $n_3$ correspond to all lines that pass through the intersection of $n_2$ and $n_3$. The dotted line illustrates one convex combination of $n_2$ and $n_3$ that pointwise dominates $n_1$: consequently, $n_1$ can be safely removed and its inward arcs re-directed to reflect this convex combination by setting the observation probabilities accordingly. In general, when a node is jointly dominated by a group of nodes, there exists a pointwise-dominating convex combination of this group.

**Theorem 1** *The value function $V(n, \cdot)$ of a node $n$ is jointly dominated by the value functions $V(n_1, \cdot), \ldots, V(n_k, \cdot)$ of nodes $n_1, \ldots, n_k$ if and only if there is a convex combination $\sum_i c_i V(n_i, \cdot)$ that dominates $V(n, \cdot)$.*

$$\min \quad \epsilon \quad \text{s.t.} \quad \sum_s b(s)V(n,s) + \epsilon \geq \sum_s b(s)V(n_i,s), \quad \forall i$$
$$\sum_s b(s) = 1; \quad b(s) \geq 0, \quad \forall s$$

Table 1: Primal LP: $V(n,\cdot)$ is jointly dominated by $V(n_1,\cdot),\ldots,V(n_k,\cdot)$ when $\epsilon \geq 0$.

$$\max \quad \epsilon \quad \text{s.t.} \quad V(n,s) + \epsilon \leq \sum_i c_i V(n_i,s), \quad \forall s \in \mathcal{S}$$
$$\sum_i c_i = 1; \quad c_i \geq 0, \quad \forall i$$

Table 2: Dual LP: convex combination $\sum_i c_i V(n_i,\cdot)$ dominates $V(n,\cdot)$ when $\epsilon \geq 0$.

**Proof:** $V(n,\cdot)$ is dominated by $V(n_1,\cdot),\ldots,V(n_k,\cdot)$ when the objective of the LP in Table 1 is positive. This LP finds the belief state $b$ that minimizes the difference between $V(n,b)$ and the max of $V(n_1,b),\ldots,V(n_k,b)$. It turns out that the dual LP (Table 2) finds the most dominating convex combination parallel to $V(n,\cdot)$. Since the dual has positive objective value when the primal does, the theorem follows. ◄

As argued in the proof of Thm. 1, the LP in Table 1 gives us an algorithm to find the most dominating convex combination parallel to a dominated node. In summary, by considering stochastic controllers, we can extend PI to prune all dominated nodes (pointwise or jointly) in the policy improvement step. This provides two advantages: controllers can be made smaller while improving their decision quality.

## 4   Bounded Policy Iteration

Although pruning all dominated nodes helps to keep the controller small, it may still grow substantially with each DP backup. Several heuristics are possible to bound the number of nodes. Feng and Hansen [6] proposed that one prunes all nodes that dominate the value function by less than some $\epsilon$ after each DP backup. Alternatively, instead of growing the controller with each backup and then pruning, we can do a *partial* DP backup that generates only a subset of the nodes using Cheng's algorithm [5], the witness algorithm [9], or other heuristics [14]. In order to keep the controller bounded, for each node created in a partial DP backup, one node must be pruned and its inward arcs redirected to some dominating convex combination. In the event where no node is dominated, we can still prune a node and redirect its arcs to a good convex combination, but the resulting controller may have lesser value at some belief states. We now propose a new algorithm called *bounded policy iteration (BPI)* that guarantees monotonic value improvement at all belief states while keeping the number of nodes fixed.

BPI considers one node at a time and tries to improve it while keeping all other nodes fixed. Improvement is achieved by replacing each node by a good convex combination of the nodes normally created by a DP backup, but without actually performing a backup. Since the backed up value function must dominate the controller's current value function, then by Thm. 1 there must exist a convex combination of the backed up nodes that point-wise dominates each node of the controller. Combining this idea with Eq. 2, we can directly compute such convex combinations with the LP in Table 3. This LP has $|\mathcal{A}||\mathcal{N}|^{|\mathcal{Z}|}$ variables corresponding to the probabilities of the convex combination as well as the $\epsilon$ variable measuring the value improvement. We can significantly reduce the number of variables by pushing the convex combination variables as far as possible into the DP backup, resulting in the LP shown in Table 4. The key here is to realize that we can aggregate many variables since we only care about the marginals $c_a = \sum_{n_1,n_2,\ldots,n_{|\mathcal{Z}|}} c_{a,n_1,n_2,\ldots,n_{|\mathcal{Z}|}}$ and $c_{a,n_z} = \sum_{n_1,\ldots,n_{z-1},n_{z+1},\ldots,n_{|\mathcal{Z}|}} c_{a,n_1,n_2,\ldots,n_{|\mathcal{Z}|}}$.

$$\begin{aligned}
\max \quad & \epsilon \\
\text{s.t.} \quad & V(n,s) + \epsilon \leq \sum_{a,n_1,n_2,\ldots,n_{|\mathcal{Z}|}} c_{a,n_1,n_2,\ldots,n_{|\mathcal{Z}|}}[R(s,a)+ \\
& \qquad \gamma \sum_{s',z} Pr(s'|s,a)Pr(z|s',a)V(n_z,s')], \ \forall s \in \mathcal{S} \\
& \sum_{a,n_1,n_2,\ldots,n_{|\mathcal{Z}|}} c_{a,n_1,n_2,\ldots,n_{|\mathcal{Z}|}} = 1; \ \ c_{a,n_1,n_2,\ldots,n_{|\mathcal{Z}|}} \geq 0, \ \forall a, n_1, n_2, \ldots, n_{|\mathcal{Z}|}
\end{aligned}$$

Table 3: Naive LP to find a convex combination of backed up nodes that dominate $n$.

$$\begin{aligned}
\max \quad & \epsilon \\
\text{s.t.} \quad & V(n,s)+\epsilon \leq \sum_a [c_a R(s,a) + \gamma \sum_{s',z} Pr(s'|s,a)Pr(z|s',a)c_{a,n_z}V(n_z,s')], \forall s \\
& \sum_a c_a = 1; \ \ \sum_{n_z} c_{a,n_z} = c_a, \ \forall a; \ \ c_a \geq 0, \ \forall a; \ \ c_{a,n_z} \geq 0, \ \forall a, z
\end{aligned}$$

Table 4: Efficient LP to find a convex combination of backed up nodes that dominate $n$.

The efficient LP in Table 4 has only $|\mathcal{A}||\mathcal{Z}||\mathcal{N}| + |\mathcal{A}| + 1$ variables.[1] Furthermore, the variables $c_a$ and $c_{a,n_z}$ have an intuitive interpretation w.r.t. the action and observation strategies for the improved node. Each $c_a$ variable indicates the probability of executing action $a$ (i.e., $\alpha(n,a) = c_a$). Similarly, each $c_{a,n_z}$ variable indicates the (unnormalized) probability of reaching node $n_z$ after executing $a$ and observing $z$ (i.e., $\beta(n,a,z,n_z) = c_{a,n_z}/c_a$). Note that we now use probabilistic action strategies and have extended probabilistic observation strategies to depend on the action executed.

To summarize, BPI alternates between policy evaluation and improvement as in regular PI, but the policy improvement step simply tries to improve each node by solving the LP in Table 4. The $c_a$ and $c_{a,n_z}$ variables are used to set the probabilistic action and observation strategies of the new improved node.

## 5  Local Optima

BPI is a simple, efficient alternative to standard PI that monotonically improves an FSC while keeping its size constant. Unfortunately, it is only guaranteed to converge to a local optimum. We now characterize BPI's local optima and propose a method to escape them.

### 5.1  Characterization

Thm. 2 gives a necessary and sufficient condition characterizing BPI's local optima. Intuitively, a controller is a local optimum when each linear segment touches from below, or is *tangent to*, the controller's backed up value function (see Fig. 1(b)).

**Theorem 2** *BPI has converged to a local optimum if and only if each node's value function is tangent to the backed up value function.*

**Proof:** Since the objective function of the LP in Table 4 seeks to maximize the improvement $\epsilon$, the resulting convex combination must be tangent to the upper surface of the backed up value function. Conversely, the only time when the LP won't be able to improve a node is when its vector is already tangent to the backed up value function. ◄

Interestingly, tangency is a necessary (but not sufficient) condition for GA's local optima.

**Corollary 1** *If GA has converged to a local optimum, then the value function of each node reachable from the initial belief state is tangent to the backed up value function.*

**Proof:** GA seeks to monotonically improve a controller in the direction of steepest ascent. The LP of Table 4 also seeks a monotonically improving direction. Thus if BPI can improve a controller by finding a direction of improvement using the LP of Table 4, then GA will also find it or will find a steeper one. Conversely, when a controller is a local optimum for GA, then there is no monotonic improvement possible in any direction. Since BPI can only improve a controller by following a direction of monotonic improvement, GA's local optima are a subset of BPI's local optima. Thus, tangency is a necessary, but not sufficient, condition of GA's local optima. ◄

In the proof of Corollary 1, we argued that GA's local optima are a subset of BPI's local optima. This suggests that BPI is inferior to GA since it can be trapped by more local optima than GA. However we will describe in the next section a simple technique that allows BPI to easily escape from local optima.

## 5.2 Escape Technique

The tangency condition characterizing local optima can be used to design an effective escape method for BPI. It essentially tells us that such tangent belief states are "bottlenecks" for further policy improvement. If we could improve the value at the tangent belief state(s) of some node, then we could break out of the local optimum. A simple method for doing so consists of a one-step lookahead search from the tangent belief states. Figure 1(b) illustrates how belief state $b'$ can be reached in one step from tangent belief state $b$, and how the backed up value function improves $b'$'s current value. Thus, if we add a node to the controller that maximizes the value of $b'$, its improved value can subsequently be backed up to the tangent belief state $b$, breaking out of the local optimum.

Our algorithm is summarized as follows: perform a one-step lookahead search from each tangent belief state; when a reachable belief state can be improved, add a new node to the controller that maximizes that belief state's value. Interestingly, when no reachable belief state can be improved, the policy must be optimal at the tangent belief states.

**Theorem 3** *If the backed up value function does not improve the value of any belief state reachable in one step from any tangent belief state, then the policy is optimal at the tangent belief states.*

**Proof:** By definition, belief states for which the backed up value function provides no improvement are tangent belief states. Hence, when all belief states reachable in one step are themselves tangent belief states, then the set of tangent belief states is closed under every policy. Since there is no possibility of improvement, the current policy must be optimal at the tangent belief states. ◄

Although Thm 3 guarantees an optimal solution only at the tangent belief states, in practice, they rarely form a proper subset of the belief space (when none of the reachable belief states can be improved). Note also that the escape algorithm assumes knowledge of the tangent belief states. Fortunately, the solution to the dual of the LP in Table 4 is a tangent belief state. Since most commercial LP solvers return both the solution of the primal and dual, a tangent belief state is readily available for each node.[2]

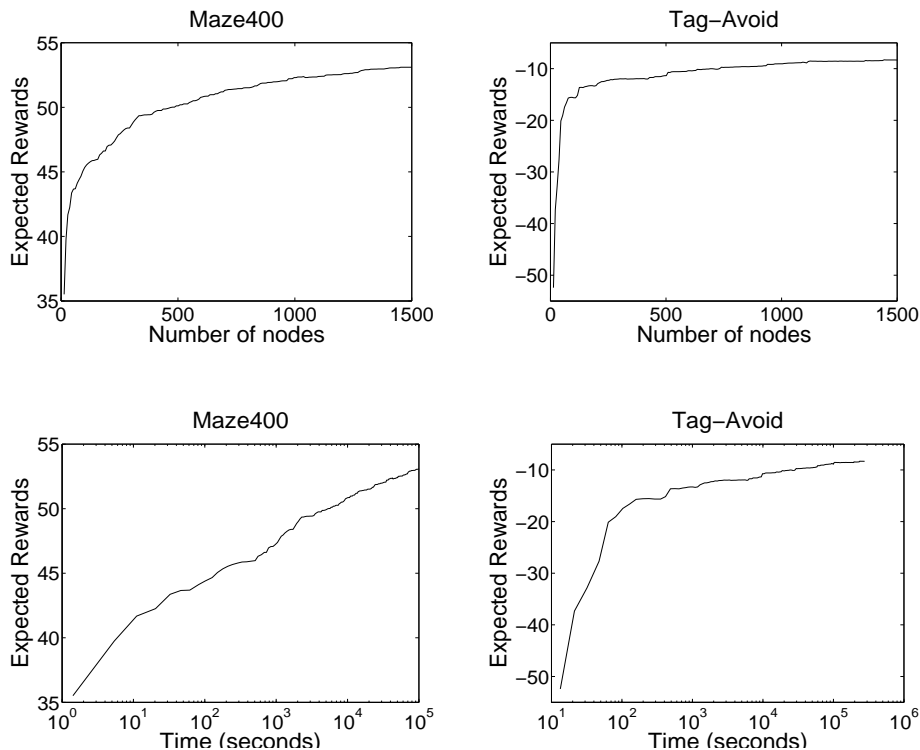

Figure 3: Experimental results for the maze and tag-avoid problems.

## 6   Experiments

We report some preliminary experiments with BPI and the escape method to assess their robustness against local optima, as well as their scalability to relatively large POMDPs. In a first experiment, we ran BPI with escape on a preference elicitation problem and a modified version of the Heaven-and-Hell problem described in [3]. It consistently found the optimal policy, whereas GA settles for a local optimum for both problems.

In a second experiment, we report the running time and decision quality of the controllers found for two large grid-world problems. The first is a 400-state extention of Hauskrecht's [8] 20-state maze problem, and the second Pineau et al.'s [12] 870-state tag-avoid problem. In Figure 3, we report the expected return achieved w.r.t. time and number of nodes. For the maze problem, the expected return is averaged over all 400 states since BPI tries to optimize the policy for all belief states simultaneously. For comparison purposes, the expected return for the tag-avoid problem is measured at the same initial belief state used in [12] even though BPI doesn't tailor its policy exclusively to that belief state. In contrast, many point-based algorithms including PBVI [12] (which is perhaps the best such algorithm) optimize the policy for a single initial belief state, capitalizing on a hopefully small reachable belief region. BPI found a 940-node controller in $59772s$ with the same expected return of $-9.18$ achieved by PBVI in $180880s$ with a policy of 1334 linear segments. This suggests that most of the belief space is reachable in tag-avoid. We also

tangent to the backed up value function, indicating that it is identical to some backed up node.

ran BPI on the tiger-grid, hallway and hallway2 benchmark problems [12] and obtained 1500-node controllers in $163420s$, $249730s$ and $274280s$ achieving expected returns of 1.81, 0.51, 0.28 at the same initial belief states used in [12], but without using them to tailor the policy. In contrast, PBVI achieved expected returns of 2.25, 0.53 and 0.34 in $3448s$, $288s$ and $360s$ with policies of 470, 86 and 95 linear segments tailored to those initial belief states. This suggests that only a small portion of the belief space is reachable.

## 7   Conclusion

We have introduced the BPI algorithm, which guarantees monotonic improvement of the value function while keeping controller size fixed. While quite efficient, the algorithm may get trapped in local optima. An analysis of such local optima reveals that the value function of each node is tangent to the backed up value function. This property can be successfully exploited in an algorithm that escapes local optima quite robustly.

This research can be extented in a number of directions. State aggregation [2] and belief compression [13] techniques could be easily integrated with BPI to scale to problems with large state spaces. Also, since stochastic GA [11, 1] can tackle model free problems (which BPI cannot) it would be interesting to see if tangent belief states could be computed for stochastic GA and used to design a heuristic to escape local optima similar to the one proposed for BPI.

**Acknowledgements** We thank Darius Braziunas for his help with the implementation and the anonymous reviewers for the helpful comments.

## Footnotes

[1]Actually, we don't need the $c_a$ variables since they can be derived from the $c_{a,n_z}$ variables by summing out $n_z$, so the number of variables can be reduced to $|\mathcal{A}||\mathcal{Z}||\mathcal{N}| + 1$.

[2]A node may have more than one tangent belief state when an interval of its linear segment is

## References

[1] D. Aberdeen and J. Baxter. Scaling internal-state policy-gradient methods for POMDPs. *Proc. ICML-02*, pp.3–10, Sydney, Australia, 2002.

[2] C. Boutilier and D. Poole. Computing optimal policies for partially observable decision processes using compact representations. *Proc. AAAI-96*, pp.1168–1175, Portland, OR, 1996.

[3] D. Braziunas. Stochastic local search for POMDP controllers. Master's thesis, University of Toronto, Toronto, 2003.

[4] A. R. Cassandra, M. L. Littman, and N. L. Zhang. Incremental pruning: A simple, fast, exact method for POMDPs. *Proc. UAI-97*, pp.54–61, Providence, RI, 1997.

[5] H.-T. Cheng. *Algorithms for Partially Observable Markov Decision Processes*. PhD thesis, University of British Columbia, Vancouver, 1988.

[6] Z. Feng and E. A. Hansen. Approximate planning for factored POMDPs. *Proc. ECP-01*, Toledo, Spain, 2001.

[7] E. A. Hansen. Solving POMDPs by searching in policy space. *Proc. UAI-98*, pp.211–219, Madison, Wisconsin, 1998.

[8] M. Hauskrecht. Value-function approximations for partially observable Markov decision processes. *Journal of Artificial Intelligence Research*, 13:33–94, 2000.

[9] L. P. Kaelbling, M. Littman, and A. R. Cassandra. Planning and acting in partially observable stochastic domains. *Artificial Intelligence*, 101:99–134, 1998.

[10] N. Meuleau, K.-E. Kim, L. P. Kaelbling, and A. R. Cassandra. Solving POMDPs by searching the space of finite policies. *Proc. UAI-99*, pp.417–426, Stockholm, 1999.

[11] N. Meuleau, L. Peshkin, K.-E. Kim, and L. P. Kaelbling. Learning finite-state controllers for partially observable environments. *Proc. UAI-99*, pp.427–436, Stockholm, 1999.

[12] J. Pineau, G. Gordon, and S. Thrun. Point-based value iteration: an anytime algorithm for POMDPs. In *Proc. IJCAI-03*, Acapulco, Mexico, 2003.

[13] P. Poupart and C. Boutilier. Value-directed compressions of POMDPs. *Proc. NIPS-02*, pp.1547–1554, Vancouver, Canada, 2002.

[14] N. L. Zhang and W. Zhang. Speeding up the convergence of value-iteration in partially observable Markov decision processes. *Journal of Artificial Intelligence Research*, 14:29–51, 2001.